# Theoretical Analysis of Learning with Reward-Modulated Spike-Timing-Dependent Plasticity

**Robert Legenstein, Dejan Pecevski, Wolfgang Maass**
Institute for Theoretical Computer Science
Graz University of Technology
A-8010 Graz, Austria
{legi,dejan,maass}@igi.tugraz.at

## Abstract

Reward-modulated spike-timing-dependent plasticity (STDP) has recently emerged as a candidate for a learning rule that could explain how local learning rules at single synapses support behaviorally relevant adaptive changes in complex networks of spiking neurons. However the potential and limitations of this learning rule could so far only be tested through computer simulations. This article provides tools for an analytic treatment of reward-modulated STDP, which allow us to predict under which conditions reward-modulated STDP will be able to achieve a desired learning effect. In particular, we can produce in this way a theoretical explanation and a computer model for a fundamental experimental finding on biofeedback in monkeys (reported in [1]).

## 1   Introduction

A major puzzle for understanding learning in biological organisms is the relationship between experimentally well-established learning rules for synapses (such as STDP) on the microscopic level and adaptive changes of the behavior of biological organisms on the macroscopic level. Neuromodulatory systems which send diffuse signals related to reinforcements (rewards) and behavioral state to several large networks of neurons in the brain, have been identified as likely intermediaries that relate these two levels of learning. It is well-known that the consolidation of changes of synaptic weights in response to pre- and postsynaptic neuronal activity requires the presence of such third signals [2]. Corresponding spike-based learning rules of the form

$$\frac{dw_{ji}(t)}{dt} = c_{ji}(t)d(t), \tag{1}$$

have been proposed in [3], where $w_{ji}$ is the weight of a synapse from neuron $i$ to neuron $j$, $c_{ji}(t)$ is an eligibility trace of this synapse which collects proposed weight changes resulting from a learning rule such as STDP, and $d(t) = h(t) - \bar{h}$ is a neuromodulatory signal with mean $\bar{h}$ (where $h(t)$ might for example represent reward prediction errors, encoded through the concentration of dopamine in the extra-cellular fluid). We will consider in this article only cases where the reward prediction error is equal to the current reward. We will refer to $d(t)$ simply as the reward signal. Obviously such learning scheme (1) faces a large credit-assignment problem, since not only those synapses for which weight changes would increase the chances of future reward receive the top-down signal $d(t)$, but billions of other synapses too. Nevertheless the brain is able to solve this credit-assignment problem, as has been shown in one of the earliest (but still among the most amazing) demonstrations of biofeedback in monkeys [1]. The spiking activity of single neurons (in area 4 of the precentral gyrus) was recorded, the current firing rate of this neuron was made visible to the monkey in the

form of an illuminated meter, and the monkey received food rewards for increases (or in alternating trials for decreases) of the firing rate of this neuron from its average level. The monkeys learnt quite reliably (on the time scale of 10's of minutes) to change the firing rate of this neuron in the currently rewarded direction[1]. Obviously the existence of learning mechanisms in the brain which are able to solve this difficult credit assignment problem is fundamental for understanding and modeling many other learning features of the brain. We present in section 3 and 4 of this abstract a learning theory for (1), where the eligibility trace $c_{ij}(t)$ results from standard forms of STDP, which is able to explain the success of the experiment in [1]. This theoretical model is confirmed by computer simulations (see section 4.1). In section 5 we leave this concrete learning experiment and investigate under what conditions neurons can learn through trial and error (via reward-modulated STDP) associations of specific firing patterns to specific patterns of input spikes. The resulting theory leads to predictions of specific parameter ranges for STDP that support this general form of learning. These were tested through computer experiments, see 5.1.

Other interesting results of computer simulations of reward-modulated STDP in the context of neural circuits were recently reported in [3] and [4] (we also refer to these articles for reviews of preceding work by Seung and others).

## 2 Models for neurons and synaptic plasticity

The spike train of a neuron $i$ which fires action potentials at times $t_i^{(1)}, t_i^{(2)}, t_i^{(3)}, \ldots$ is formalized by a sum of Dirac delta functions $S_i(t) = \sum_{t_i^{(n)}} \delta(t - t_i^{(n)})$. We assume that positive and negative weight changes suggested by STDP for all pairs of pre- and postsynaptic spikes (according to the two integrals in (2)) are collected in an eligibility trace $c_{ji}(t)$, where the impact of a spike pairing with the second spike at time $t - s$ on the eligibility trace at time $t$ is given by some function $f_c(s)$ for $s \geq 0$:

$$c_{ji}(t) = \int_0^\infty ds f_c(s) \left[ \int_0^\infty dr \, W(r) S_j^{post}(t-s) S_i^{pre}(t-s-r) \right.$$
$$\left. + \int_0^\infty dr \, W(-r) S_j^{post}(t-s-r) S_i^{pre}(t-s) \right]. \quad (2)$$

In our simulations, $f_c(s)$ is a function of the form $f_c(s) = \frac{s}{\tau_e} e^{-\frac{s}{\tau_e}}$ if $s \geq 0$ and 0 otherwise, with time constant $\tau_e = 0.5$s. $W(r)$ denotes the standard exponential STDP learning window

$$W(r) = \begin{cases} A_+ e^{-r/\tau_+} & , \quad \text{if } r \geq 0 \\ -A_- e^{r/\tau_-} & , \quad \text{if } r < 0 \end{cases}, \quad (3)$$

where the positive constants $A_+$ and $A_-$ scale the strength of potentiation and depression, $\tau_+$ and $\tau_-$ are positive time constants defining the width of the positive and negative learning window, and $S_i^{pre}, S_j^{post}$ are the spike trains of the presynaptic and postsynaptic neuron respectively. The actual weight change is the product of the eligibility trace with the reward signal as defined by equation (1). We assume that weights are clipped at the lower boundary value 0 and an upper boundary $w_{max}$.

We use a linear Poisson neuron model whose output spike train $S_j^{post}(t)$ is a realization of a Poisson process with the underlying instantaneous firing rate $R_j(t)$. The effect of a spike of presynaptic neuron $i$ at time $t'$ on the membrane potential of neuron $j$ is modeled by an increase in the instantaneous firing rate by an amount $w_{ji}(t')\epsilon(t - t')$, where $\epsilon$ is a response kernel which models the time course of a postsynaptic potential (PSP) elicited by an input spike. Since STDP according to [3] has been experimentally confirmed only for excitatory synapses, we will consider plasticity only for excitatory connections and assume that $w_{ji} \geq 0$ for all $i$ and $\epsilon(s) \geq 0$ for all $s$. Because the synaptic response is scaled by the synaptic weights, we can assume without loss of generality that the response kernel is normalized to $\int_0^\infty ds \, \epsilon(s) = 1$. In this linear model, the contributions of all inputs are summed up linearly:

$$R_j(t) = \sum_{i=1}^n \int_0^\infty ds \, w_{ji}(t-s) \, \epsilon(s) \, S_i(t-s) \,, \quad (4)$$

where $S_1, \ldots, S_n$ are the $n$ presynaptic spike trains.

## 3  Theoretical analysis of the resulting weight changes

We are interested in the expected weight change over some time interval $T$ (see [5]), where the expectation is over realizations of the stochastic input- and output spike trains as well as a stochastic realization of the reward signal, denoted by the ensemble average $\langle \cdot \rangle_E$

$$\frac{\langle w_{ji}(t+T) - w_{ji}(t)\rangle_E}{T} = \frac{1}{T}\left\langle \int_t^{t+T} \frac{d}{dt}w_{ji}(t')dt' \right\rangle_E = \left\langle \left\langle \frac{d}{dt}w_{ji}(t)\right\rangle_T \right\rangle_E, \quad (5)$$

where we used the abbreviation $\langle f(t)\rangle_T = T^{-1}\int_t^{t+T} f(t')\,dt'$. Using equation (1), this yields

$$\frac{\langle w_{ji}(t+T) - w_{ji}(t)\rangle_E}{T} = \int_0^\infty dr\, W(r)\int_0^\infty ds\, f_c(s)\, \langle D_{ji}(t,s,r)\, \nu_{ji}(t-s,r)\rangle_T$$
$$+ \int_{-\infty}^0 dr\, W(r)\int_{|r|}^\infty ds\, f_c(s+r)\, \langle D_{ji}(t,s,r)\, \nu_{ji}(t-s,r)\rangle_T, (6)$$

where $D_{ji}(t,s,r) = \langle d(t)|$ Neuron $j$ spikes at $t-s$, and neuron $i$ spikes at $t-s-r\rangle_E$ is the average reward at time $t$ given a presynaptic spike at time $t-s-r$ and a postsynaptic spike at time $t-s$, and $\nu_{ji}(t,r) = \langle S_j(t)S_i(t-r)\rangle_E$ describes correlations between pre- and postsynaptic spike timings (see [6] for the derivation). We see that the expected weight change depends on how the correlations between the pre- and postsynaptic neurons correlate with the reward signal. If these correlations are varying slowly with time, we can exploit the self-averaging property of the weight vector. Analogously to [5], we can drop the ensemble average on the left hand side and obtain:

$$\frac{d}{dt}\langle w_{ji}(t)\rangle_T = \int_0^\infty dr\, W(r)\int_0^\infty ds\, f_c(s)\, \langle D_{ji}(t,s,r)\, \nu_{ji}(t-s,r)\rangle_T$$
$$+ \int_{-\infty}^0 dr\, W(r)\int_{|r|}^\infty ds\, f_c(s+r)\, \langle D_{ji}(t,s,r)\, \nu_{ji}(t-s,r)\rangle_T. \quad (7)$$

In the following, we will always use the smooth time-averaged vector $\langle w_{ji}(t)\rangle_T$, but for brevity, we will drop the angular brackets. If one assumes for simplicity that the impact of a pre-post spike pair on the eligibility trace is always triggered by the postsynaptic spike, one gets (see [6] for details):

$$\frac{dw_{ji}(t)}{dt} = \int_0^\infty ds\, f_c(s)\int_{-\infty}^\infty dr\, W(r)\, \langle D_{ji}(t,s,r)\, \nu_{ji}(t-s,r)\rangle_T. \quad (8)$$

This assumption (which is common in STDP analysis) will introduce a small error for post-before pre spike pairs, since if a reward signal arrives at some time $d_r$ after the pairing, the weight update will be proportional to $f_c(d_r)$ instead of $f_c(d_r + r)$. For the analyses presented in this article, the simplified equation (8) is a good approximation for the learning dynamics (see [6]). Equation (8) shows that if the reward signal does not depend on pre- and postsynaptic spike statistics, the weight will change according to standard STDP scaled by a constant proportional to the mean reward.

## 4  Application to biofeedback experiments

We now apply our theoretical approach to the biofeedback experiments by Fetz and Baker [1] that we have sketched in the introduction. The authors showed that it is possible to increase and decrease the firing rate of a randomly chosen neuron by rewarding the monkey for its high (respectively low) firing rates. We assume in our model that a reward is delivered to *all* neurons in the simulated recurrent network with some delay $d_r$ every time a specific neuron $k$ in the network produces an action potential

$$d(t) = \int_0^\infty dr\, S_k^{post}(t-d_r-r)\epsilon_r(r). \quad (9)$$

where $\epsilon_r(r)$ is the shape of the reward pulse corresponding to one postsynaptic spike of the reinforced neuron. We assume that the reward kernel $\epsilon_r$ has zero mass, i.e., $\bar\epsilon_r = \int_0^\infty dr\, \epsilon_r(r) = 0$. In

our simulations, this reward kernel will have a positive bump in the first few hundred milliseconds, and a long tailed negative bump afterwards. With the linear Poisson neuron model (see Section 2), the correlation of the reward with pre-post spike pairs of the reinforced neuron is (see [6])

$$D_{ki}(t,s,r) = w_{ki} \int_0^\infty dr' \; \epsilon_r(r')\epsilon(s+r-d_r-r') + \epsilon_r(s-d_r) \approx \epsilon_r(s-d_r). \qquad (10)$$

The last approximation holds if the impact of a single input spike on the membrane potential is small. The correlation of the reward with pre-post spike pairs of non-reinforced neurons is

$$D_{ji}(t,s,r) = \int_0^\infty dr' \; \epsilon_r(r') \frac{\nu_{kj}(t-d_r-r',s-d_r-r') + w_{ki}w_{ji}\epsilon(s+r-d_r-r')\epsilon(r)}{\nu_j(t-s) + w_{ji}\epsilon(r)}. \qquad (11)$$

If the contribution of a single postsynaptic potential to the membrane potential is small, we can neglect the impact of the presynaptic spike and write

$$D_{ji}(t,s,r) \approx \int_0^\infty dr' \; \epsilon_r(r') \frac{\nu_{kj}(t-d_r-r',s-d_r-r')}{\nu_j(t-s)}. \qquad (12)$$

Hence, the reward-spike correlation of a non-reinforced neuron depends on the correlation of this neuron with the reinforced neuron. The mean weight change for weights to the reinforced neuron is given by

$$\frac{d}{dt}w_{ki}(t) = \int_0^\infty ds \; f_c(s+d_r)\epsilon_r(s) \int_{-\infty}^\infty dr \; W(r) \left\langle \nu_{ki}(t-d_r-s,r) \right\rangle_T. \qquad (13)$$

This equation basically describes STDP with a learning rate that is proportional to the eligibility function in the time around the reward-delay. The mean weight change of neurons $j \neq k$ is given by

$$\frac{d}{dt}w_{ji}(t) = \int_0^\infty ds \; f_c(s) \int_{-\infty}^\infty dr \; W(r) \int_0^\infty dr'\epsilon_r(r') \left\langle \frac{\nu_{kj}(t-d_r-r',s-d_r-r')}{\nu_j(t-s)} \nu_{ji}(t-s,r) \right\rangle_T \qquad (14)$$

If the output of neurons $j$ and $k$ are uncorrelated, this evaluates to approximately zero (see [6]).

The result can be summarized as follows. The reinforced neuron is trained by STDP. Other neurons are trained by STDP with a learning rate proportional to their correlation with the reinforced neuron. If a neuron is uncorrelated with the reinforced neuron, the learning rate is approximately zero.

## 4.1 Computer simulations

In order to test the theoretical predictions for the experiment described in the previous section, we have performed a computer simulation with a generic neural microcircuit receiving a global reward signal. This global reward signal increases its value every time a specific neuron (the reinforced neuron) in the circuit fires. The circuit consists of 1000 leaky integrate-and-fire (LIF) neurons (80% excitatory and 20% inhibitory), which are interconnected by conductance based synapses. The short term dynamics of synapses was modeled in accordance with experimental data (see [6]). Neurons within the recurrent circuit were randomly connected with probabilities $p_{ee} = 0.08$, $p_{ei} = 0.08$, $p_{ie} = 0.096$ and $p_{ii} = 0.064$ where the ee, ei, ie, ii indices designate the type of the presynaptic and postsynaptic neurons (excitatory or inhibitory). To reproduce the synaptic background activity of neocortical neurons in vivo, an Ornstein-Uhlenbeck (OU) conductance noise process modeled according to ([7]) was injected in the neurons, which also elicited spontaneous firing of the neurons in the circuit with an average rate of 4Hz. In half of the neurons part of the noise was substituted with random synaptic connections from the circuit, in order to observe how the learning mechanisms work when most of the input conductance in the neuron comes from a larger number of input synapses which are plastic, instead of a static noise process. The function $f_c(t)$ from equation (2) had the form $f_c(t) = \frac{t}{\tau_e}e^{-\frac{t}{\tau_e}}$ if $t \geq 0$ and 0 otherwise, with time constant $\tau_e = 0.5$s. The reward signal during the simulation was computed according to eq. (9), with the following shape for $\epsilon_r(t)$

$$\epsilon_r(t) = A_r^+ \frac{t}{\tau_r^+} e^{-\frac{t}{\tau_r^+}} - A_r^- \frac{t}{\tau_r^-} e^{-\frac{t}{\tau_r^-}}. \qquad (15)$$

The parameter values for $\epsilon_r(t)$ were chosen such as to produce a positive reward pulse with a peak delayed 0.5s from the spike that caused it, and a long tailed negative bump so that $\int_0^\infty dt \; \epsilon_r(t) = 0$.

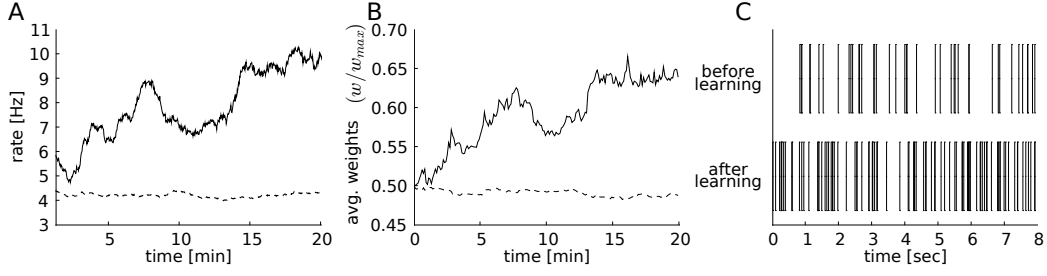

Figure 1: Computer simulation of the experiment by Fetz and Baker [1]. **A)** The firing rate of the reinforced neuron (solid line) increases while the average firing rate of 20 other randomly chosen neurons in the circuit (dashed line) remains unchanged. **B)** Evolution of the average synaptic weight of excitatory synapses connecting to the reinforced neuron (solid line) and to other neurons (dashed line). **C)** Spike trains of the reinforced neuron at the beginning and at the end of the simulation.

For values of other model parameters see [6]. The learning rule (1) was applied to all synapses in the circuit which have excitatory presynaptic and postsynaptic neurons. The simulation was performed for 20 min simulated biological time with a simulation time step of 0.1ms.

Fig. 1 shows that the firing rate and synaptic weights of the reinforced neuron increase within a few minutes of simulated biological time, while those of the other neurons remain largely unchanged. Note that this reinforcement learning task is more difficult than that of the first computer experiment of [3], where postsynaptic firing within 10 ms after presynaptic firing of a randomly chosen synapse was rewarded, since the relationship between synaptic activity (and hence with STDP) is less direct in this setup. Whereas a very low spontaneous firing rate of 1 Hz was required in [3], this simulation shows that reinforcement learning is also feasible at rate levels which correspond to those reported in [1].

## 5   Rewarding spike-timings

In order to explore the limits of reward-modulated STDP, we have also investigated a substantially more demanding reinforcement learning scenario. The reward signal $d(t)$ was given in dependence on how well the output spike train $S_j^{post}$ of the neuron $j$ matched some rather arbitrary spike train $S^*$ that was produced by some neuron that received the same $n$ input spike trains as the trained neuron with arbitrary weights $\mathbf{w}^* = (w_1^*, \ldots, w_n^*)^T$, $w_i^* \in \{0, w_{max}\}$, but in addition $n' - n$ further spike trains $S_{n+1}, \ldots, S_{n'}$ with weights $w_i^* = w_{max}$. This setup provides a generic reinforcement learning scenario, when a quite arbitrary (and not perfectly realizable) spike output is reinforced, but simultaneously the performance of the learner can be evaluated quite clearly according to how well its weights $w_1, \ldots, w_n$ match those of the target neuron for those $n$ input spike trains which both of them receive. The reward $d(t)$ at time $t$ is given by

$$d(t) = \int_{-\infty}^{\infty} dr \, \kappa(r) S_j^{post}(t - d_r) S^*(t - d_r - r), \tag{16}$$

where the function $\kappa(r)$ with $\bar{\kappa} = \int_{-\infty}^{\infty} ds \, \kappa(s) > 0$ describes how the reward signal depends on the time difference between a postsynaptic spike and a target spike and $d_r > 0$ is the delay of the reward. Our theoretical analysis below suggests that this reinforcement learning task can in principle be solved by reward-modulated STDP if some constraints are fulfilled. The analysis also reveals which reward kernels $\kappa$ are suitable for this learning setup. The reward correlation for synapse $i$ is (see [6])

$$D_{ji}(t, s, r) = \int_{-\infty}^{\infty} dr' \kappa(r') \left[ \nu_j^{post}(t - d_r) + \delta(s - d_r) + w_{ji}(s + r - d_r)\epsilon(s + r - d_r) \right]$$

$$\left[ \nu^*(t - d_r - r') + w_i^* \epsilon(s + r - d_r - r') \right], \quad (17)$$

where $\nu_j^{post}(t) = \langle S_j^{post}(t) \rangle_E$ denotes the mean rate of the trained neuron at time $t$, and $\nu^*(t) = \langle S^*(t) \rangle_E$ denotes the mean rate of the target spike train at time $t$. Since weights are changing very

slowly, we have $w_{ji}(t - s - r) = w_{ji}(t)$. In the following, we will drop the dependence of $w_{ji}$ on $t$ for brevity. For simplicity, we assume that input rates are stationary and uncorrelated. In this case (since the weights are changing slowly), also the correlations between inputs and outputs can be assumed stationary, $\nu_{ji}(t, r) = \nu_{ji}(r)$. We assume that the eligibility function $f_c(d_r) \approx f_c(d_r + r)$ if $|r|$ is on a time scale of a PSP, the learning window, or the reward kernel, and that $d_r$ is large compared to these time scales. Then, for uncorrelated Poisson input spike trains of rate $\nu_i^{pre}$ and the linear Poisson neuron model, the weight change at synapse $ji$ is given by

$$
\begin{aligned}
\frac{dw_{ji}(t)}{dt} \approx{}& \bar{\kappa}\bar{f}_c\nu^*\nu_i^{pre}\nu_j^{post}\left[\nu_j^{post}\bar{W} + w_{ji}\bar{W}_\epsilon\right]\\
&+\bar{\kappa}f_c(d_r)\nu_i^{pre}\left[\nu_j^{post}\bar{W} + w_{ji}\bar{W}_\epsilon\right]\left[\nu^* + \nu^* w_{ji} + w_i^* \nu_j^{post}\right]\\
&+f_c(d_r)w_i^*\nu_i^{pre}\left[\nu_j^{post}\int_{-\infty}^{\infty}dr\,W(r)\epsilon_\kappa(r) + w_{ji}\int_{-\infty}^{\infty}dr\,W(r)\epsilon(r)\epsilon_\kappa(r)\right]\\
&+f_c(d_r)w_i^*w_{ji}\nu_i^{pre}\left[\nu_j^{post}\bar{W} + w_{ji}\bar{W}_\epsilon\right]\int_0^{\infty}dr\,\epsilon(r)\epsilon_\kappa(r),
\end{aligned}
\tag{18}
$$

where $\bar{f}_c = \int_0^\infty dr\, f_c(r)$, $\bar{W} = \int_{-\infty}^\infty dr\, W(r)$, $\epsilon_\kappa(r) = \int_{-\infty}^\infty dr'\, \kappa(r')\epsilon(r - r')$ is the convolution of the reward kernel with the PSP is the integral over the STDP learning window, and $\bar{W}_\epsilon = \int_{-\infty}^\infty dr\, \epsilon(r)W(r)$.

We will now bound the expected weight change for synapses $ji$ with $w_i^* = w_{max}$ and for synapses $jk$ with $w_{jk}^* = 0$. In this way we can derive conditions for which the expected weight change for the former synapses is positive, and that for the latter type is negative. First, we assume that the integral over the reward kernel is positive. In this case, the weight change is negative for synapses $i$ with $w_i^* = 0$ if and only if $\nu_i^{pre} > 0$, and $-\nu_j^{post}\bar{W} > w_{ji}\bar{W}_\epsilon$. In the worst case, $w_{ji}$ is $w_{max}$ and $\nu_j^{post}$ is small. We have to guarantee some minimal output rate $\nu_{min}^{post}$ such that even if $w_{ji} = w_{max}$, this inequality is fulfilled. This could be guaranteed by some noise current. For synapses $i$ with $w_i^* = w_{max}$, we obtain two more conditions (see [6] for a derivation). The conditions are summarized in inequalities (19)-(21). If these inequalities are fulfilled and input rates are positive, then the weight vector converges on average from any initial weight vector to $\mathbf{w}^*$.

$$
-\nu_{min}^{post}\bar{W} > w_{max}\bar{W}_\epsilon \tag{19}
$$

$$
\int_{-\infty}^{\infty}dr\,W(r)\epsilon(r)\epsilon_\kappa(r) \geq -\nu_{max}^{post}\bar{W}\int_0^{\infty}dr\,\epsilon(r)\epsilon_\kappa(r) \tag{20}
$$

$$
\int_{-\infty}^{\infty}dr\,W(r)\epsilon_\kappa(r) > -\bar{W}\bar{\kappa}\left[\frac{\nu^*\nu_{max}^{post}}{w_{max}}\frac{\bar{f}_c}{f_c(d_r)} + \frac{\nu^*}{w_{max}} + \nu^* + \nu_{max}^{post}\right], \tag{21}
$$

where $\nu_{max}^{post}$ is the maximal output rate. The second condition is less severe, and should be easily fulfilled in most setups. If this is the case, the first condition (19) ensures that weights with $w^* = 0$ are depressed while the third condition (21) ensures that weights with $w^* = w_{max}$ are potentiated.

**Optimal reward kernels:** From condition (21), we can deduce optimal reward kernels $\kappa$. The kernel should be such that the integral $\int_{-\infty}^\infty dr\, W(r)\epsilon_\kappa(r)$ is large, while the integral over $\kappa$ is small (but positive). Hence, $\epsilon_\kappa(r)$ should be positive for $r > 0$ and negative for $r < 0$. In the following experiments, we use a simple kernel which satisfies the aforementioned constraints:

$$
\kappa(r) = \begin{cases} A_+^\kappa\left(e^{-\frac{t-t_\kappa}{\tau_1^\kappa}} - e^{-\frac{t-t_\kappa}{\tau_2^\kappa}}\right) & , \quad \text{if } t - t_\kappa \geq 0\\ -A_-^\kappa\left(e^{\frac{t-t_\kappa}{\tau_1^\kappa}} - e^{\frac{t-t_\kappa}{\tau_2^\kappa}}\right) & , \quad \text{otherwise} \end{cases}
$$

where $A_+^\kappa$ and $A_-^\kappa$ are positive scaling constants, $\tau_1^\kappa$ and $\tau_2^\kappa$ define the shape of the two double-exponential functions the kernel is composed of, and $t_\kappa$ defines the offset of the zero-crossing from the origin. The optimal offset from the origin is negative and in the order of tens of milliseconds for usual PSP-shapes $\epsilon$. Hence, reward is positive if the neuron spikes around the target spike or somewhat later, and negative if the neuron spikes much too early.

## 5.1 Computer simulations

In the computer simulations we explored the learning rule in a more biologically realistic setting, where we used a leaky integrate-and-fire (LIF) neuron with input synaptic connections coming from

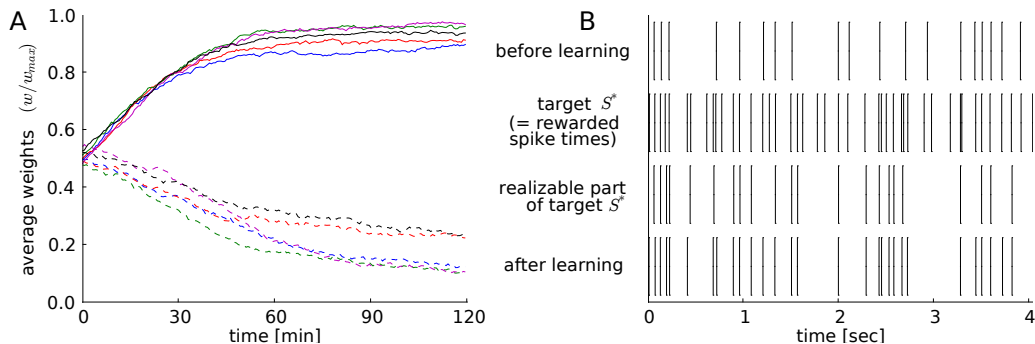

Figure 2: Reinforcement learning of spike times. **A)** Synaptic weight changes of the trained LIF neuron, for 5 different runs of the experiment. The curves show the average of the synaptic weights that should converge to $w_i^* = 0$ (dashed lines), and the average of the synaptic weights that should converge to $w_i^* = w_{max}$ (solid lines) with a different shading for each simulation run. **B)** Comparison of the output of the trained neuron before (upper trace) and after learning (lower trace; the same input spike trains and the same noise inputs were used before and after training for 2 hours). The second trace from above shows those spike times which are rewarded, the third trace shows the target spike train without the additional noise inputs.

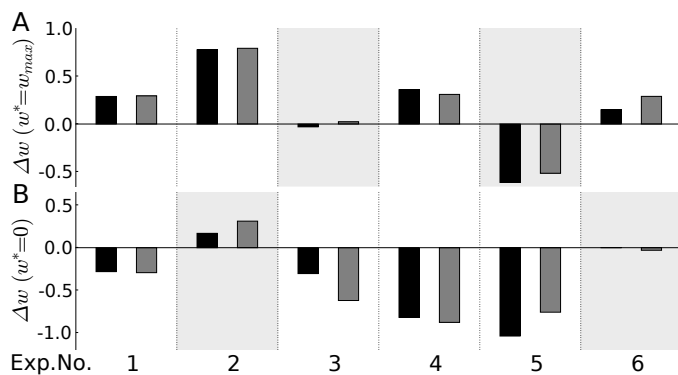

Figure 3: Predicted average weight change (black bars) calculated from equation (18), and the estimated average weight change (gray bars) from simulations, presented for 6 different experiments with different parameter settings (see Table 1).[2] **A)** Weight change values for synapses with $w_i^* = w_{max}$. **B)** Weight change values for synapses with $w_i^* = 0$. Cases where the constraints are not fulfilled are shaded with gray color.

a generic neural microcircuit composed of 1000 LIF neurons. The synapses were conductance based exhibiting short term facilitation and depression. The trained neuron and the arbitrarily given neuron which produced the target spike train $S^*$ ("target neuron") both were connected to the same randomly chosen, 100 excitatory and 10 inhibitory neurons from the circuit. The target neuron had 10 additional excitatory input connections (these weights were set to $w_{max}$), not accessible to the trained neuron. Only the synapses of the trained neuron connecting from excitatory neurons were set to be plastic. The target neuron had a weight vector with $w_i^* = 0$ for $0 \leq i < 50$ and $w_i^* = w_{max}$ for $50 \leq i < 110$. The generic neural microcircuit from which the trained and the target neurons receive the input had 80% excitatory and 20% inhibitory neurons interconnected randomly with a probability of 0.1. The neurons received background synaptic noise as modeled in [7], which caused spontaneous activity of the neurons with an average firing rate of 6.9Hz. During the simulations, we observed a firing rate of 10.6Hz for the trained, and 19Hz for the target neuron. The reward was delayed by 0.5s, and we used the same eligibility trace function $f_c(t)$ as in the simulations for the biofeedback experiment (see [6] for details). The simulations were run for two hours simulated biological time, with a simulation time step of 0.1ms. We performed 5 repetitions of the experiment, each time with different randomly generated circuits and different initial weight values for the trained neuron. In each of the 5 runs, the average synaptic weights of synapses with $w_i^* = w_{max}$ and $w_i^* = 0$ approach their target values, as shown in Fig. 2A. In order to test how

| Ex. | $\tau_\epsilon$[ms] | $w_{max}$ | $\nu_{min}^{post}$ [Hz] | $A_+10^6$ | $\frac{A_-}{A_+}$ | $\tau_+,\tau_2^\kappa$ [ms] | $A_+^\kappa$ | $t_{sim}$ [h] |
|---|---|---|---|---|---|---|---|---|
| 1 | 10 | 0.012 | 10 | 16.62 | 1.05 | 20,20 | 3.34 | 5 |
| 2 | 7 | 0.020 | 5 | 11.08 | 1.02 | 15,16 | 4.58 | 10 |
| 3 | 20 | 0.010 | 6 | 5.54 | 1.10 | 25,40 | 1.46 | 16 |
| 4 | 7 | 0.020 | 5 | 11.08 | 1.07 | 25,16 | 4.67 | 13 |
| 5 | 10 | 0.015 | 6 | 20.77 | 1.10 | 25,20 | 3.75 | 3 |
| 6 | 25 | 0.005 | 3 | 13.85 | 1.01 | 25,20 | 3.34 | 13 |

Table 1: Parameter values used for the simulations in Figure 3. Both cases where the constraints are satisfied and not satisfied were covered. PSPs were modeled as $\epsilon(s) = e^{(-s/\tau_\epsilon)}/\tau_\epsilon$.

closely the learning neuron reproduces the target spike train $S^*$ after learning, we have performed additional simulations where the same spiking input $S_I$ is applied to the learning neuron before and after we conducted the learning experiment (results are reported in Fig. 2B).

The equations in section 5 define a parameter space for which the trained neuron can learn the target synapse pattern $\mathbf{w}^*$. We have chosen 6 different parameter values encompassing cases with satisfied and non-satisfied constraints, and performed experiments where we compare the predicted average weight change from equation (18) with the actual average weight change produced by simulations. Figure 3 summarizes the results. In all 6 experiments, the sufficient conditions (19)-(21) were correct. In those cases where these conditions were not met, the weight moved in the opposite direction, suggesting that the theoretically sufficient conditions (19)-(21) might also be necessary.

## 6  Discussion

We have developed in this paper a theory of reward-modulated STDP. This theory predicts that reinforcement learning through reward-modulated STDP is also possible at biologically more realistic spontaneous firing rates than the average rate of 1 Hz that was used (and argued to be needed) in the extensive computer experiments of [3]. We have also shown both analytically and through computer experiments that the result of the fundamental biofeedback experiment in monkeys from [1] can be explained on the basis of reward-modulated STDP. The resulting theory of reward-modulated STDP makes concrete predictions regarding the shape of various functions (e.g. reward functions) that would optimally support the speed of reward-modulated learning for the generic (but rather difficult) learning tasks where a neuron is supposed to respond to input spikes with specific patterns of output spikes, and only spikes at the right times are rewarded. Further work (see [6]) shows that reward-modulated STDP can in some cases replace supervised training of readout neurons from generic cortical microcircuit models.

**Acknowledgment:** We would like to thank Gordon Pipa and Matthias Munk for helpful discussions. Written under partial support by the Austrian Science Fund FWF, project # P17229, project # S9102 and project # FP6-015879 (FACETS) of the European Union.

## Footnotes

[1]Adjacent neurons tended to change their firing rate in the same direction, but also differential changes of directions of firing rates of pairs of neurons are reported in [1] (when these differential changes were rewarded).

[2]The values in the figure are calculated as $\Delta w = \frac{\overline{w}(t_{sim}) - \overline{w}(0)}{w_{max}/2}$ for the simulations, and with $\Delta w = \frac{\langle dw/dt \rangle t_{sim}}{w_{max}/2}$ for the predicted value. $\overline{w}(t)$ is the average weight over synapses with the same value of $w^*$.

## References

[1] E. E. Fetz and M. A. Baker. Operantly conditioned patterns of precentral unit activity and correlated responses in adjacent cells and contralateral muscles. *J Neurophysiol*, 36(2):179–204, Mar 1973.

[2] C. H. Bailey, M. Giustetto, Y.-Y. Huang, R. D. Hawkins, and E. R. Kandel. Is heterosynaptic modulation essential for stabilizing Hebbian plasticity and memory? *Nature Reviews Neuroscience*, 1:11–20, 2000.

[3] E. M. Izhikevich. Solving the distal reward problem through linkage of STDP and dopamine signaling. *Cerebral Cortex Advance Access*, January 13:1–10, 2007.

[4] R. V. Florian. Reinforcement learning through modulation of spike-timing-dependent synaptic plasticity. *Neural Computation*, 6:1468–1502, 2007.

[5] W. Gerstner and W. M. Kistler. *Spiking Neuron Models*. Cambridge University Press, Cambridge, 2002.

[6] R. Legenstein, D. Pecevski, and W. Maass. Theory and applications of reward-modulated spike-timing-dependent plasticity. *in preparation*, 2007.

[7] J.M. Fellous A. Destexhe, M. Rudolph and T.J. Sejnowski. Fluctuating synaptic conductances recreate in vivo-like activity in neocortical neurons. *Neuroscience*, 107(1):13–24, 2001.